# Neural Network Analysis of Distributed Representations of Dynamical Sensory-Motor Transformations in the Leech

**Shawn R. Lockery, Yan Fang, and Terrence J. Sejnowski**
Computational Neurobiology Laboratory
Salk Institute for Biological Studies
Box 85800, San Diego, CA 92138

## ABSTRACT

Interneurons in leech ganglia receive multiple sensory inputs and make synaptic contacts with many motor neurons. These "hidden" units coordinate several different behaviors. We used physiological and anatomical constraints to construct a model of the local bending reflex. Dynamical networks were trained on experimentally derived input-output patterns using recurrent back-propagation. Units in the model were modified to include electrical synapses and multiple synaptic time constants. The properties of the hidden units that emerged in the simulations matched those in the leech. The model and data support distributed rather than localist representations in the local bending reflex. These results also explain counterintuitive aspects of the local bending circuitry.

## INTRODUCTION

Neural network modeling techniques have recently been used to predict and analyze the connectivity of biological neural circuits (Zipser and Andersen, 1988; Lehky and Sejnowski, 1988; Anastasio and Robinson, 1989). Neurons are represented as simplified processing units and arranged into model networks that are then trained to reproduce the input-output function of the reflex or brain region of interest. After training, the receptive and projective field of hidden units in the network often bear striking similarities to actual neurons and can suggest functional roles of neurons with inputs and outputs that are hard to grasp intuitively. We applied this approach to the local bending reflex of the leech, a three-layered, feed-forward network comprising a small number of identifiable

neurons whose connectivity and input-output function have been determined physiologically. We found that model local bending networks trained using recurrent back-propagation (Pineda, 1987; Pearlmutter, 1989) to reproduce a physiologically determined input-output function contained hidden units whose connectivity and temporal response properties closely resembled those of identified neurons in the biological network. The similarity between model and actual neurons suggested that local bending is produced by distributed representations of sensory and motor information.

## THE LOCAL BENDING REFLEX

In response to a mechanical stimulus, the leech withdraws from the site of contact (Fig. 1a). This is accomplished by contraction of longitudinal muscles beneath the stimulus and relaxation of longitudinal muscles on the opposite side of the body, resulting in a U-shaped local bend (Kristan, 1982). The form of the response is independent of the site of stimulation: dorsal, ventral, and lateral stimuli produce an appropriately oriented

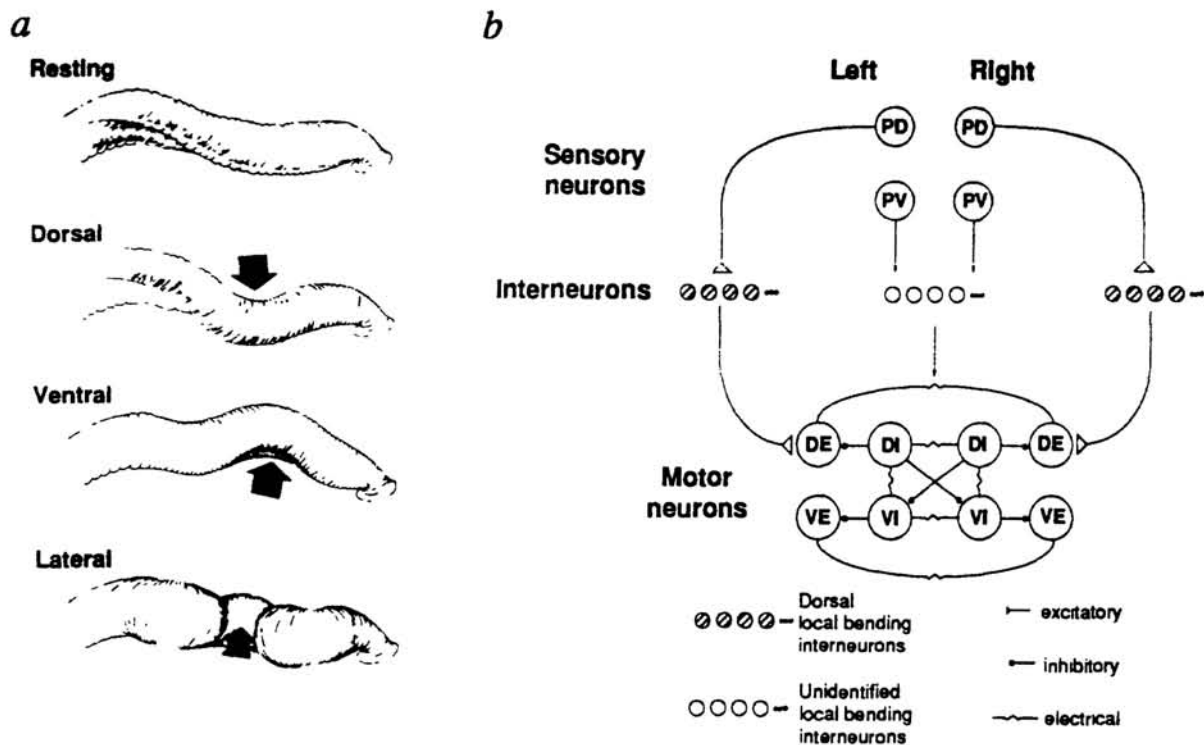

**Figure 1:** *a.* Local bending behavior. Partial view of a leech in the resting position and in response to dorsal, ventral, and lateral stimuli. *b.* Local bending circuit. The main input to the reflex is provided by the dorsal and ventral P cells (PD and PV). Control of local bending is largely provided by motor neurons whose field of innervation is restricted to single left-right, dorsal-ventral quadrants of the body; dorsal and ventral quadrants are innervated by both excitatory (DE and VE) and inhibitory (DI and VI) motor neurons. Motor neurons are connected by electrical and chemical synapses. Sensory input to motor neurons is mediated by a layer of interneurons. Interneurons that were excited by PD and which in turn excite DE have been identified (hatched) ; other types of interneurons remain to be identified (open).

withdrawal. Major input to the local bending reflex is provided by four pressure sensitive mechanoreceptors called P cells, each with a receptive field confined to a single quadrant of the body wall (Fig. 1b). Output to the muscles is provided by eight types of longitudinal muscle motor neurons, one to four excitatory and inhibitory motor neurons for each body wall quadrant (Stuart, 1970; Ort et al., 1974). Motor neurons are connected by chemical and electrical synapses that introduce the possibility of feedback among the motor neurons.

Dorsal, ventral, and lateral stimuli each produce a pattern of P cell activation that results in a unique pattern of activation and inhibition of the motor neurons (Lockery and Kristan, 1990a). Connections between sensory and motor neurons are mediated by a layer of interneurons (Kristan, 1982). Nine types of local bending interneurons have been identified (Lockery and Kristan, 1990b). These comprise the subset of the local bending interneurons which contribute to dorsal local bending because they are excited by the dorsal P cell and in turn excite the dorsal excitatory motor neuron. There appear to be no functional connections between interneurons. Other interneurons remain to be identified, such as those which *inhibit* the dorsal excitatory motor neurons.

Interneuron input connections were determined by recording the amplitude of the postsynaptic potential in an interneuron while each of the P cells was stimulated with a standard train of impulses (Lockery and Kristan, 1990b). Output connections were determined by recording the amplitude of the postsynaptic potential in each motor neuron when an interneuron was stimulated with a standard current pulse. Interneuron input and output connections are shown in Figure 2, where white squares are excitatory connections, black squares are inhibitory connections, and the size of each square indicates connection strength. Most interneurons received substantial input from three or four P cells, indicating that the local bending network forms a distributed representation of sensory input.

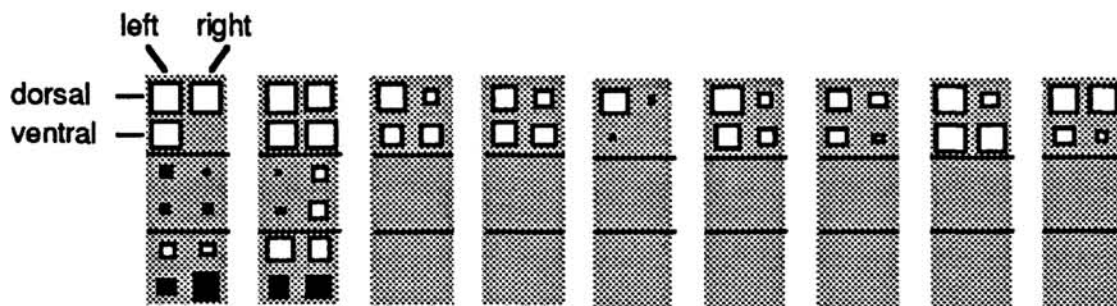

**Figure 2:** Input and output connections of the nine types of dorsal local bending interneurons. Within each gray box, the upper panel shows input connections from sensory neurons, the middle panel shows output connections to inhibitory motor neurons, and the lower panel shows output connections to excitatory motor neurons. Side-length of each box is proportional to the amplitude of the connection determined from intracellular recordings of interneurons or motor neurons. White boxes indicate excitatory connections and black boxes indicated inhibitory connections. Blank spaces denote conections whose strength has not been determined for technical reasons.

## NEURAL NETWORK MODEL

Because sensory input is represented in a distributed fashion, most interneurons are active in all forms of local bending. Thus, in addition to contributing to *dorsal* local bending, most interneurons are also active during *ventral* and *lateral* bending when some or all of their output effects are inappropriate to the observed behavioral response. This suggests that the inappropriate effects of the dorsal bending interneurons must be offset by other as yet unidentified interneurons and raises the possibility that local bending is the result of simultaneous activation of a population of interneurons with multiple sensory inputs and both appropriate and inappropriate effects on many motor neurons. It was not obvious, however, that such a population was sufficient, given the well-known nonlinearities of neural elements and constraints imposed by the input-output function and connections known to exist in the network. The possibility remained that interneurons specific for each form of the behavior were required to produce each output pattern. To address this issue, we used recurrent back-propagation (Pearlmutter, 1989) to train a dynamical network of model neurons (Fig 3a). The network had four input units representing the

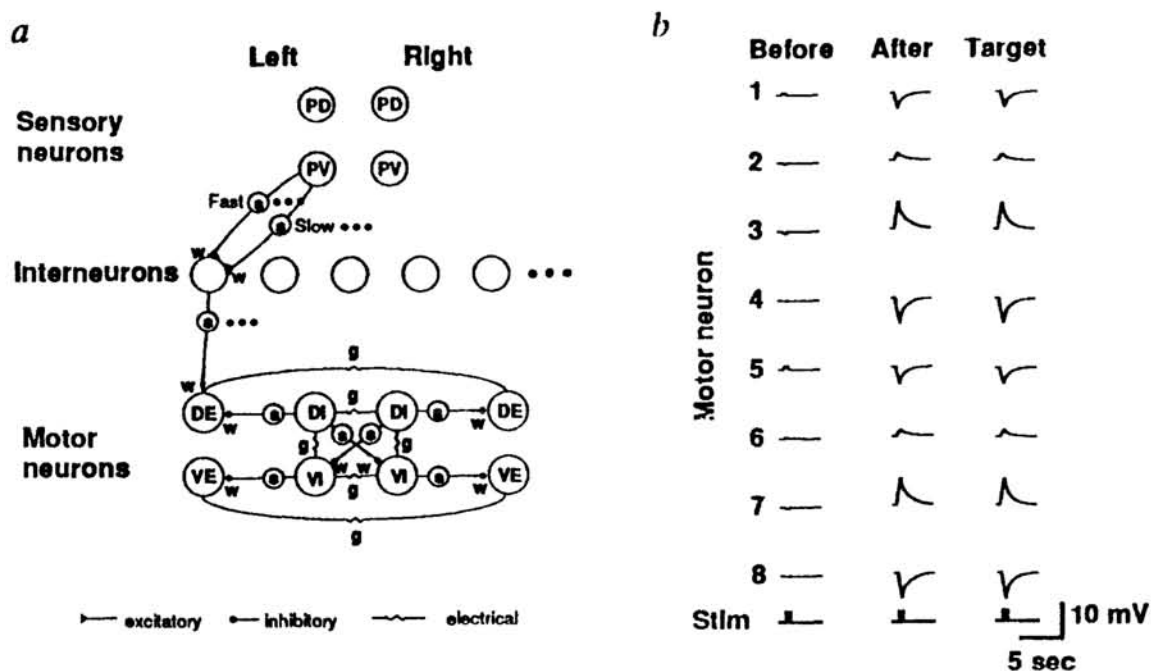

Figure 3: *a*. The local bending network model. Four sensory neurons were connected to eight motor neurons via a layer of 10 interneurons. Neurons were represented as single electrical compartments whose voltage varied as a function of time (see text). Known electrical and chemical connections among motor neurons were assigned fixed connection strengths (g's and w's) determined from intracellular recordings. Interneuron input and output connections were adjusted by recurrent back-propagation. Chemical synaptic delays were implemented by inserting s-units between chemically connected pairs of neurons. S-units with different time constants were inserted between sensory and interneurons to account for fast and slow components of synaptic potentials recorded in interneurons. *b*. Output of the model network in response to simultaneous activation of both PDs (stim). The response of each motor neuron (rows) is shown before and after training. The desired response contained in the training set is shown on the right for comparison (target).

four P cells, and eight output units representing the eight motor neuron types. Between input and output units was a single layer of 10 hidden units representing the interneurons. Neurons were represented as single electrical compartments with an input resistance and time constant. The membrane potential ($V_i$) of each neuron was given by

$$T_i \, dV_i/dt = -V_i + R_i(I_e + I_c)$$

where $T_i$ and $R_i$ are the time constant and input resistance of the neuron and $I_e$ and $I_c$ are the sum of the electrical and chemical synaptic currents from presynaptic neurons. Current due to electrical synapses was given by

$$I_e = \Sigma_j \, g_{ij} \, (V_j\text{-}V_i)$$

where $g_{ij}$ is the coupling conductance between neuron i and j. To implement the delay associated with chemical synapses, synapse units (s-units) were inserted between between pairs of neurons connected by chemical synapses. The activation of each s-unit was given by

$$T_{ij} \, dS_{ij}/dt = -S_{ij} + f(V_j)$$

where $T_{ij}$ is the synaptic time constant and $f(V_j)$ was a physiologically determined sigmoidal function ($0 \le f \le 1$) relating pre- and postsynaptic membrane potential at an identified monosynaptic connection in the leech (Granzow et al., 1985). Current due to chemical synapses was given by

$$I_c = \Sigma_j \, w_{ij} \, S_{ij}$$

where $w_{ij}$ is the strength of the chemical synapse between units i and j. Thus, synaptic current is a graded function of presynaptic voltage, a common feature of neurons in the leech (Friesen, 1985; Granzow et al., 1985; Thompson and Stent, 1976) and other invertebrates (Katz and Miledi, 1967; Burrows and Siegler, 1978; Nagayama and Hisada, 1987).

Chemical and electrical synaptic strengths between motor neurons were determined by recording from pairs of motor neurons and were not adjusted by the training algorithm. Interneuron input and output connections were given small initial values that were randomly assigned and subsequently adjusted during training. During training, input connections were constrained to be positive to reflect the fact that only excitatory interneuron input connections were seen (Fig. 2), but no constraints were placed on the number of input or output connections. Synaptic time constants were assigned fixed values. These were adjusted by hand to fit the time course of motor neuron synaptic potentials (Lockery and Kristan, 1990a), or determined from pairwise motor neuron recordings (Granzow et al., 1985).

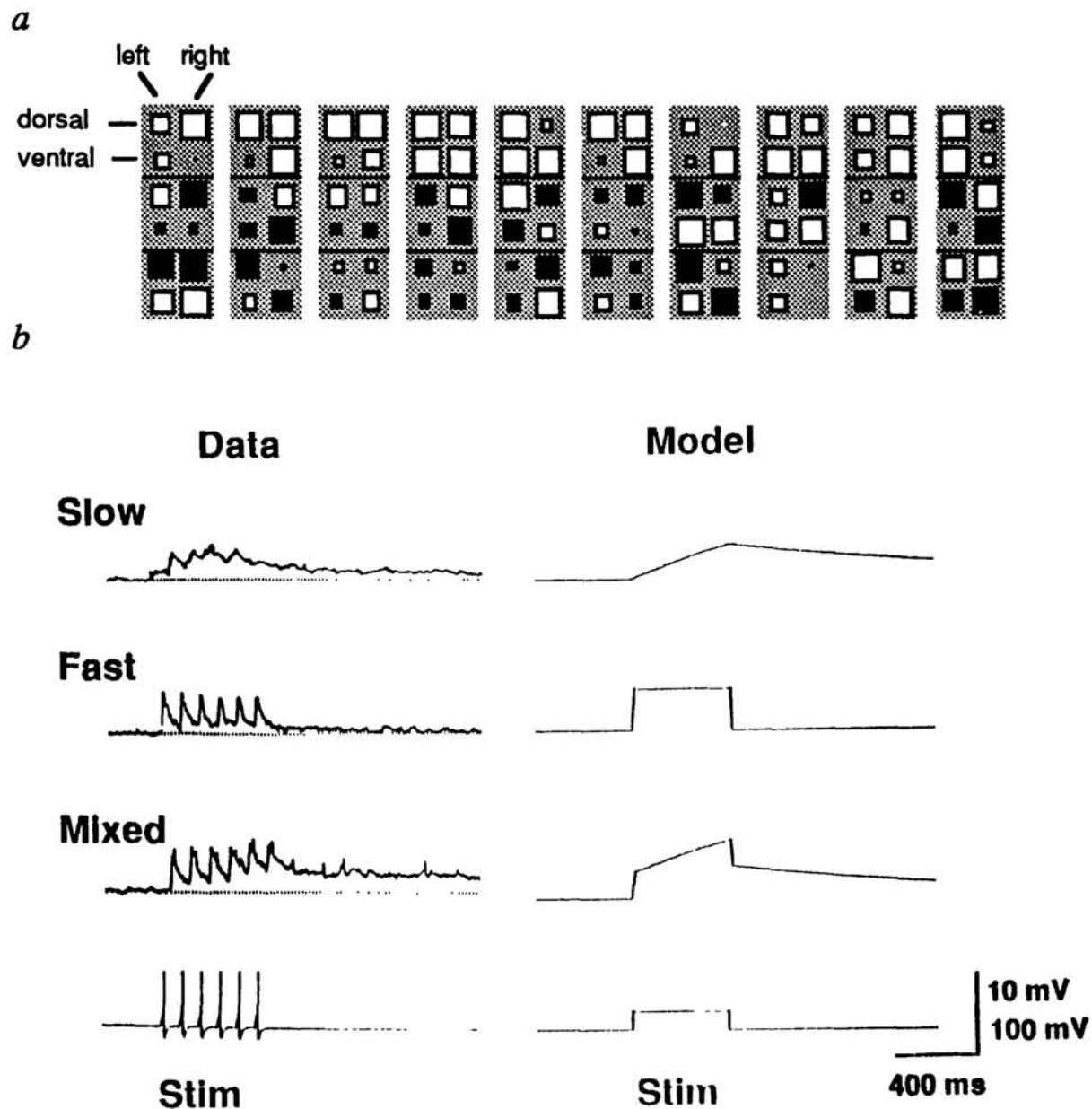

**Figure 4:** *a.* Input and output connections of model local bending interneurons. Model interneurons, like the actual interneurons, received substantial inputs from three or four sensory neurons and had significant effects on most of the motor neurons. Symbols as in figure 2. *b.* Actual (data) and simulated (model) synaptic potentials recorded from three types of interneuron. Actual synaptic potentials were recorded in response to a train of P cell impulses. Simulated synaptic potentials were recorded in response to a pulse of current in the P cell which simulates a step change in P cell firing frequency.

## RESULTS

Model networks were trained to produce the amplitude and time course of synaptic potentials recorded in all eight motor neurons in response to trains of P cell impulses

(Lockery and Kristan, 1990a). The training set included the response of all eight motor neurons when each P cell was stimulated alone and when P cells were stimulated in pairs. After 6,000 - 10,000 training epochs, the output of the model closely matched the desired output for all patterns in the training set (Fig. 3b). To compare interneurons in the model network to actual interneurons, simulated physiological experiments were performed. Interneuron input connections were determined by recording the amplitude of the postsynaptic potential in a model interneuron while each of the P cells was stimulated with a standard current pulse. Output connections were determined by recording the amplitude of the postsynaptic potential in each motor neuron when an interneuron was stimulated with a standard current pulse. Model interneurons, like those in the real network, received three or four substantial connections from P cells and had significant effects on most of the motor neurons (Fig. 4a). Most model interneurons were active during each form of the behavior and the output connections of the interneurons were only partially consistent with each form of the local bending response. Thus, the appropriate motor neuron responses were produced by the summation of many appropriate and inappropriate interneuron effects. This result explains the appropriate and inappropriate effects of interneurons in the leech.

There was also agreement between the time course of the response of model and actual interneurons to P cell stimulation (Fig. 4b). In the actual network, interneuron synaptic potentials in response to trains of P cell impulses had a fast and slow component. Some interneurons showed only the fast component, some only the slow, and some showed both components (mixed). Although no constraints were placed on the temporal response properties of interneurons, the same three types of interneuron were found in the model network. The three different types of interneuron temporal response were due to different relative connection strengths of fast and slow s-units impinging on a given interneuron (Fig. 3a).

## CONCLUSION

Our results show that the network modeling approach can be adapted to models with more realistic neurons and synaptic connections, including electrical connections, which occur in both invertebrates and vertebrates. The qualitative similarity between model and actual interneurons demonstrates that a population of interneurons resembling the identified dorsal local bending interneurons could mediate local bending in a distributed processing system without additional interneurons specific for different forms of local bending. Interneurons in the model also displayed the diversity in temporal responses seen in interneurons in the leech. Clearly, the training algorithm did not produce exact matches between model and actual interneurons, but this was not surprising since the identified local bending interneurons represent only a subset of the interneurons in the reflex. More exact matches could be obtained by using two pools of model interneurons, one to represent identified neurons, the other to represent unidentified neurons. Model neurons in the latter pool would constitute testable physiological predictions of the connectivity of unidentified local bending interneurons.

### Acknowledgements

Supported by the Bank of America-Giannini Foundation, the Drown Foundation, and the Mathers Foundation.

# References

Anastasio, T. and Robinson, D. A. (1989) Distributed parallel processing in the vestibulo-oculomotor system. Neural Comp. 1:230-241.

Burrows, M., and M.V.S. Siegler (1978) Graded synaptic transmission between local interneurones and motor neurones in the metathoracic ganglion of the locust. J. Physiol. 285:231-255.

Friesen, W.O. (1985) Neuronal control of leech swimming movements: interactions between cell 60 and previously described oscillator neurons. J. Comp. Physiol. 156:231-242.

Granzow, B., W.O. Friesen, and W.B. Kristan Jr. (1985) Physiological and morphological analysis of synaptic transmission between leech motor neurons. J.Neurosci. 5:2035-2050.

Katz, B., and Miledi, R. (1967) Synaptic transmission in the absence of nerve impulses. J. Physiol. 192:407-436.

Kristan Jr., W.B. (1982) Sensory and motor neurons responsible for the local bending response in leeches. J. Exp. Biol. 96:161-180.

Kristan, W.B. Jr., S.J. McGirr, and G.V. Simpson (1982) Behavioral and mechanosensory neurone responses to skin stimulation in leeches. J. Exp. Biol. 96:143-160.

Lehky, S.R., and T.J. Sejnowski (1988) Network model of shape-from-shading: neural function arises from both receptive and projective fields. Nature 333:452-454.

Lockery, S.R., and W.B. Kristan Jr. (1990) Distributed processing of sensory information in the leech. I. Input-output relations of the local bending reflex. J. Neurosci. (in press).

Lockery, S.R., and W.B. Kristan Jr. (1990) Distributed processing of sensory information in the leech. II. Identification of interneurons contributing to the local bending reflex. J. Neurosci. (in press).

Nagayama, T., and M. Hisada (1987) Opposing parallel connections through crayfish local nonspiking interneurons. J. Comp. Neurol. 257:347-358.

Nicholls, J.G., and D. Purves (1970) Monosynaptic chemical and electrical connexions between sensory and motor cells in the central nervous system of the leech. J. Physiol. 209:647-667.

Nicholls, J.G., and B.G. Wallace (1978) Quantal analysis of transmitter release an in inhibitory synapse in the CNS. J. Physiol. 281:157-170.

Ort, C.A., W.B. Kristan Jr., and G.S. Stent (1974) Neuronal control of swimming in the medicinal leech. II. Identification and connections of motor neurones. J. Comp. Physiol. 94:121-154.

Stuart, A.E. (1970) Physiological and morphological properties of motoneurones in the central nervous system of the leech. J. Physiol. 209:627-646.

Thompson, W.J., and G.S. Stent (1976) Neuronal control of heartbeat in the medicinal leech. J. Comp. Physiol. 111:309-333.

Zipser, D., and R.A. Andersen (1988) A back-propagation programmed network that simulates response properties of a subset of posterior parietal neurons Nature 331:679-684.